# Probabilistic latent variable models for distinguishing between cause and effect

**Joris M. Mooij**
MPI for Biological Cybernetics
Tübingen, Germany
joris.mooij@tuebingen.mpg.de

**Oliver Stegle**
MPI for Biological Cybernetics
Tübingen, Germany
oliver.stegle@tuebingen.mpg.de

**Dominik Janzing**
MPI for Biological Cybernetics
Tübingen, Germany
dominik.janzing@tuebingen.mpg.de

**Kun Zhang**
MPI for Biological Cybernetics
Tübingen, Germany
kun.zhang@tuebingen.mpg.de

**Bernhard Schölkopf**
MPI for Biological Cybernetics
Tübingen, Germany
bernhard.schoelkopf@tuebingen.mpg.de

## Abstract

We propose a novel method for inferring whether $X$ causes $Y$ or vice versa from joint observations of $X$ and $Y$. The basic idea is to model the observed data using probabilistic latent variable models, which incorporate the effects of unobserved noise. To this end, we consider the hypothetical effect variable to be a *function* of the hypothetical cause variable and an *independent* noise term (not necessarily additive). An important novel aspect of our work is that we do not restrict the model class, but instead put general non-parametric priors on this function and on the distribution of the cause. The causal direction can then be inferred by using standard Bayesian model selection. We evaluate our approach on synthetic data and real-world data and report encouraging results.

## 1 Introduction

The challenge of inferring whether $X$ causes $Y$ ("$X \to Y$") or vice versa ("$Y \to X$") from joint observations of the pair $(X, Y)$ has recently attracted increasing interest [1, 2, 3, 4, 5, 6, 7, 8]. While the traditional causal discovery methods [9, 10] based on (conditional) independences between variables require at least three observed variables, some recent approaches can deal with pairs of variables by exploiting the *complexity* of the (conditional) probability distributions. On an intuitive level, the idea is that the factorization of the joint distribution $P(\text{cause}, \text{effect})$ into $P(\text{cause})P(\text{effect} \mid \text{cause})$ typically yields models of lower total complexity than the factorization into $P(\text{effect})P(\text{cause} \mid \text{effect})$. Although the notion of "complexity" is intuitively appealing, it is not obvious how it should be precisely defined.

If complexity is measured in terms of Kolmogorov complexity, this kind of reasoning would be in the spirit of the principle of "algorithmically independent conditionals" [11], which can also be embedded into a general theory of algorithmic-information-based causal discovery [12]. The following theorem is implicitly stated in the latter reference (see remarks before (26) therein):

**Theorem 1** *Let $P(X,Y)$ be a joint distribution with finite Kolmogorov complexity such that $P(X)$ and $P(Y \mid X)$ are algorithmically independent, i.e.,*

$$I\big(P(X) : P(Y \mid X)\big) \stackrel{\pm}{=} 0 \,, \tag{1}$$

*where $\stackrel{\pm}{=}$ denotes equality up to additive constants. Then:*

$$K\big(P(X)\big) + K\big(P(Y \mid X)\big) \stackrel{+}{\leq} K\big(P(Y)\big) + K\big(P(X \mid Y)\big) \,. \tag{2}$$

The proof is given by observing that (1) implies that the shortest description of $P(X,Y)$ is given by separate descriptions of $P(X)$ and $P(Y \mid X)$. It is important to note at this point that the total complexity of the causal model consists of both the complexity of the conditional distribution and of the marginal of the putative cause. However, since Kolmogorov complexity is uncomputable, this does not solve the causal discovery problem in practice. Therefore, other notions of complexity need to be considered.

The work of [4] measures complexity in terms of norms in a reproducing kernel Hilbert space, but due to the high computational costs it applies only to cases where one of the variables is binary. The methods [1, 2, 3, 5, 6] define classes of conditionals $\mathcal{C}$ and marginal distributions $\mathcal{M}$, and prefer $X \to Y$ whenever $P(X) \in \mathcal{M}$ and $P(Y \mid X) \in \mathcal{C}$ but $P(Y) \notin \mathcal{M}$ or $P(X \mid Y) \notin \mathcal{C}$. This can be interpreted as a (crude) notion of model complexity: all probability distributions inside the class are simple, and those outside the class are complex. However, this *a priori* restriction to a particular class of models poses serious practical limitations (even when in practice some of these methods "soften" the criteria by, for example, using the $p$-values of suitable hypothesis tests).

In the present work we propose to use a fully non-parametric, Bayesian approach instead. The key idea is to define appropriate priors on marginal distributions (of the cause) and on conditional distributions (of the effect given the cause) that both favor distributions of low complexity. To decide upon the most likely causal direction, we can compare the marginal likelihood (also called evidence) of the models corresponding to each of the hypotheses $X \to Y$ and $Y \to X$. An important novel aspect of our work is that we explicitly treat the "noise" as a latent variable that summarizes the influence of all other unobserved causes of the effect. The additional key assumption here is the independence of the "causal mechanism" (the function mapping from the cause and noise to the effect) and the distribution of the cause, an idea that was exploited in a different way recently for the deterministic (noise-free) case [13]. The three main contributions of this work are:

- to show that causal discovery for the two-variable cause-effect problem can be done without restricting the class of possible causal mechanisms;
- to point out the importance of accounting for the complexity of the distribution of the cause, in addition to the complexity of the causal mechanism (like in equation (2));
- to show that a Bayesian approach can be used for causal discovery even in the case of two continuous variables, without the need for explicit independence tests.

The last aspect allows for a straightforward extension of the method to the multi-variable case, the details of which are beyond the scope of this article.[1] Apart from discussing the proposed method on a theoretical level, we also evaluate our approach on both simulated and real-world data and report good empirical results.

## 2 Theory

We start with a theoretical treatment of how to solve the basic causal discovery task (see Figure 1a).

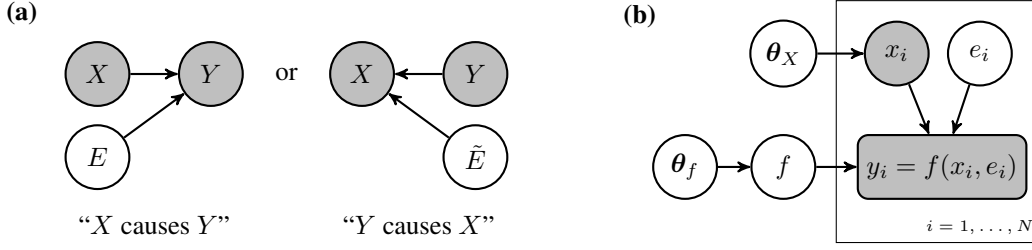

Figure 1: Observed variables are colored gray, and unobserved variables are white. **(a)** The basic causal discovery task: which of the two causal models gives the best explanation of the observed data $\mathcal{D} = \{(x_i, y_i)\}_{i=1}^N$? **(b)** More detailed version of the graphical model for "$X$ causes $Y$".

## 2.1 Probabilistic latent variable models for causal discovery

First, we give a more precise definition of the class of models that we use for representing that $X$ causes $Y$ ("$X \rightarrow Y$"). We assume that the relationship between $X$ and $Y$ is not deterministic, but disturbed by unobserved noise $E$ (effectively, the summary of all other unobserved causes of $Y$). The situation is depicted in the left-hand part of Figure 1a: $X$ and $E$ both cause $Y$, but although $X$ and $Y$ are observed, $E$ is not. We make the following additional assumptions:

(A) There are no other causes of $Y$, or in other words, we assume *determinism*: a function $f$ exists such that
$$Y = f(X, E).$$
This function will henceforth be called the *causal mechanism*.

(B) $X$ and $E$ have no common causes, i.e., $X$ and $E$ are independent:
$$X \perp\!\!\!\perp E.$$

(C) The distribution of the cause is "independent" from the causal mechanism.[2]

(D) The noise has a standard-normal distribution: $E \sim \mathcal{N}(0, 1)$.[3]

Several recent approaches to causal discovery are based on the assumptions (A) and (B) only, but pose one of the following additional restrictions on $f$:

- $f$ is linear [2];
- *additive noise* [5], where $f(X, E) = F(X) + E$ for some function $F$ ;
- the *post-nonlinear model* [6], where $f(X, E) = G(F(X) + E)$ for some functions $F, G$.

For these special cases, it has been shown that a model of the same (restricted) form in the reverse direction $Y \rightarrow X$ that induces the same joint distribution on $(X, Y)$ does *not* exist in general. This asymmetry can be used for inferring the causal direction.

In practice, a limited model class may lead to wrong conclusions about the causal direction. For example, when assuming additive noise, it may happen that neither of the two directions provides a sufficiently good fit to the data and hence no decision can be made. Therefore, we would like to drop this kind of assumptions that limit the model class. However, assumptions (A) and (B) are not enough on their own: in general, one can always construct a random variable $\tilde{E} \sim \mathcal{N}(0, 1)$ and a function $\tilde{f} : \mathbb{R}^2 \rightarrow \mathbb{R}$ such that
$$X = \tilde{f}(Y, \tilde{E}), \qquad Y \perp\!\!\!\perp \tilde{E} \tag{3}$$
(for a proof of this statement, see e.g., [14, Theorem 1]).

In combination with the other two assumptions (C) and (D), however, one does obtain an asymmetry that can be used to infer the causal direction. Note that assumption (C) still requires a suitable mathematical interpretation. One possibility would be to interpret this independence as an algorithmic

independence similar to Theorem 1, but then we could not use it in practice. Another interpretation has been used in [13] for the noise-free case (i.e., the deterministic model $Y = f(X)$). Here, our aim is to deal with the noisy case. For this setting we propose a Bayesian approach, which will be explained in the next subsection.

## 2.2 The Bayesian generative model for $X \to Y$

The basic idea is to define non-parametric priors on the causal mechanisms and input distributions that favor functions and distributions of low complexity. Inferring the causal direction then boils down to standard Bayesian model selection, where preference is given to the model with the largest marginal likelihood.

We introduce random variables $x_i$ (the cause), $y_i$ (the effect) and $e_i$ (the noise), for $i = 1, \ldots, N$ where $N$ is the number of data points. We use vector notation $\boldsymbol{x} = (x_i)_{i=1}^N$ to denote the whole $N$-tuple of $X$-values $x_i$, and similarly for $\boldsymbol{y}$ and $\boldsymbol{e}$. To make a Bayesian model comparison between the two models $X \to Y$ and $Y \to X$, we need to calculate the marginal likelihoods $p(\boldsymbol{x}, \boldsymbol{y} \mid X \to Y)$ and $p(\boldsymbol{x}, \boldsymbol{y} \mid Y \to X)$. Below, we will only consider the model $X \to Y$ and omit this from the notation for brevity. The other model $Y \to X$ is completely analogous, and can be obtained by simply interchanging the roles of $X$ and $Y$.

The marginal likelihood for the observed data $\boldsymbol{x}, \boldsymbol{y}$ under the model $X \to Y$ is given by (see also Figure 1b):

$$p(\boldsymbol{x}, \boldsymbol{y}) = p(\boldsymbol{x})p(\boldsymbol{y} \mid \boldsymbol{x}) =$$

$$\left[ \int \left( \prod_{i=1}^N p(x_i \mid \boldsymbol{\theta}_X) \right) p(\boldsymbol{\theta}_X) d\boldsymbol{\theta}_X \right] \left[ \int \left( \prod_{i=1}^N \delta\big(y_i - f(x_i, e_i)\big) p_E(e_i) \right) d\boldsymbol{e} \, p(f \mid \boldsymbol{\theta}_f) df \, p(\boldsymbol{\theta}_f) d\boldsymbol{\theta}_f \right] \tag{4}$$

Here, $\boldsymbol{\theta}_X$ and $\boldsymbol{\theta}_f$ parameterize prior distributions of the cause $X$ and the causal mechanism $f$, respectively. Note how the four assumptions discussed in the previous subsection are incorporated into the model: assumption (A) results in Dirac delta distributions $\delta\big(y_i - f(x_i, e_i)\big)$ for each $i = 1, \ldots, N$. Assumption (B) is realized by the *a priori* independence $p(\boldsymbol{x}, \boldsymbol{e} \mid \boldsymbol{\theta}_X) = p(\boldsymbol{x} \mid \boldsymbol{\theta}_X) p_{\boldsymbol{E}}(\boldsymbol{e})$. Assumption (C) is realized as the *a priori* independence $p(f, \boldsymbol{\theta}_X) = p(f)p(\boldsymbol{\theta}_X)$. Assumption (D) is obvious by taking $p_E(e) := \mathcal{N}(e \mid 0, 1)$.

## 2.3 Choosing the priors

In order to completely specify the model $X \to Y$, we need to choose particular priors. In this work, we assume that all variables are real numbers (i.e., $\boldsymbol{x}$, $\boldsymbol{y}$ and $\boldsymbol{e}$ are random variables taking values in $\mathbb{R}^N$), and use the following choices (although other choices are also possible):

- For the prior distribution of the cause $X$, we use a Gaussian mixture model

$$p(x_i \mid \boldsymbol{\theta}_X) = \sum_{j=1}^k \alpha_j \mathcal{N}(x_i \mid \mu_j, \sigma_j^2)$$

  with hyperparameters $\boldsymbol{\theta}_X = (k, \alpha_1, \ldots, \alpha_k, \mu_1, \ldots, \mu_k, \sigma_1, \ldots, \sigma_k)$. We put an improper Dirichlet prior (with parameters $(-1, -1, \ldots, -1)$) on the component weights $\boldsymbol{\alpha}$ and flat priors on the component parameters $\boldsymbol{\mu}, \boldsymbol{\sigma}$.
- For the prior distribution $p(f \mid \boldsymbol{\theta}_f)$ of the causal mechanism $f$, we take a Gaussian process with zero mean function and squared-exponential covariance function:

$$k_{\boldsymbol{\theta}_f}\big((x, e), (x', e')\big) = \lambda_Y^2 \exp\left(-\frac{(x - x')^2}{2\lambda_X^2}\right) \exp\left(-\frac{(e - e')^2}{2\lambda_E^2}\right) \tag{5}$$

  where $\boldsymbol{\theta}_f = (\lambda_X, \lambda_Y, \lambda_E)$ are length-scale parameters. The parameter $\lambda_Y$ determines the amplitude of typical functions $f(x, e)$, and the length scales $\lambda_X$ and $\lambda_E$ determine how quickly typical functions change depending on $x$ and $e$, respectively. In the additive noise case, for example, the length scale $\lambda_E$ is large compared to the length scale $\lambda_X$, as this leads to an almost linear dependence of $f$ on $e$. We put broad Gamma priors on all length-scale parameters.

## 2.4 Approximating the evidence

Now that we have fully specified the model $X \to Y$, the remaining task is to calculate the integral (4) for given observations $\boldsymbol{x}, \boldsymbol{y}$. As the exact calculation seems intractable, we here use a particular approximation of this integral.

**The marginal distribution**

For the model of the distribution of the cause $p(\boldsymbol{x})$, we use an asymptotic expansion based on the Minimum Message Length principle that yields the following approximation (for details, see [15]):

$$-\log p(\boldsymbol{x}) \approx \min_{\boldsymbol{\theta}_X} \left( \sum_{j=1}^{k} \log \left( \frac{N\alpha_j}{12} \right) + \frac{k}{2} \log \frac{N}{12} + \frac{3k}{2} - \log p(\boldsymbol{x} \,|\, \boldsymbol{\theta}_X) \right). \tag{6}$$

**The conditional distribution**

For the conditional distribution $p(\boldsymbol{y} \,|\, \boldsymbol{x})$ according to the model $X \to Y$, we start by replacing the integral over the length-scales $\boldsymbol{\theta}_f$ by a MAP estimate:

$$p(\boldsymbol{y} \,|\, \boldsymbol{x}) \approx \max_{\boldsymbol{\theta}_f} p(\boldsymbol{\theta}_f) \int \delta\big(\boldsymbol{y} - f(\boldsymbol{x}, \boldsymbol{e})\big) \, p_{\boldsymbol{E}}(\boldsymbol{e}) d\boldsymbol{e} \, p(f \,|\, \boldsymbol{\theta}_f) df.$$

Integrating over the latent variables $\boldsymbol{e}$ and using the Dirac delta function calculus (where we assume invertability of the functions $f_x : e \mapsto f(e, x)$ for all $x$), we obtain:[4]

$$\int \delta\big(\boldsymbol{y} - f(\boldsymbol{x}, \boldsymbol{e})\big) \, p_{\boldsymbol{E}}(\boldsymbol{e}) d\boldsymbol{e} \, p(f \,|\, \boldsymbol{\theta}_f) df = \int p_{\boldsymbol{E}}\big(\boldsymbol{\epsilon}(f)\big) \frac{p(f \,|\, \boldsymbol{\theta}_f)}{J(f)} \, df \tag{7}$$

where $\boldsymbol{\epsilon}(f)$ is the (unique) vector satisfying $\boldsymbol{y} = f(\boldsymbol{x}, \boldsymbol{\epsilon})$, and

$$J(f) = \det \big|\nabla_{\boldsymbol{e}} f\big(\boldsymbol{x}, \boldsymbol{\epsilon}(f)\big)\big| = \prod_{i=1}^{N} \left| \frac{\partial f}{\partial e}\big(x_i, \epsilon_i(f)\big) \right|$$

is the absolute value of the determinant of the Jacobian which results when integrating over the Dirac delta function. The next step would be to integrate over all possible causal mechanisms $f$ (which would be an infinite-dimensional integral). However, this integral again seems intractable, and hence we revert to the following approximation. Because of space constraints, we only give a brief sketch of the procedure here.

Let us suppress the hyperparameters $\boldsymbol{\theta}_f$ for the moment to simplify notation. The idea is to approximate the infinite-dimensional GP function $f$ by a linear combination over basis functions $\phi_j$ parameterized by a weight vector $\boldsymbol{\alpha} \in \mathbb{R}^N$ with a Gaussian prior distribution:

$$f_{\boldsymbol{\alpha}}(x, e) = \sum_{j=1}^{N} \alpha_j \phi_j(x, e), \qquad \boldsymbol{\alpha} \sim \mathcal{N}(\mathbf{0}, \mathbf{1}).$$

Now, defining the matrix $\Phi_{ij}(\boldsymbol{x}, \boldsymbol{\epsilon}) := \phi_j(x_i, \epsilon_i)$, the relationship $\boldsymbol{y} = \boldsymbol{\Phi}(\boldsymbol{x}, \boldsymbol{\epsilon})\boldsymbol{\alpha}$ gives a correspondence between $\boldsymbol{\epsilon}$ and $\boldsymbol{\alpha}$ (for fixed $\boldsymbol{x}$ and $\boldsymbol{y}$), which we assume to be one-to-one. In particular, $\boldsymbol{\alpha} = \Phi(\boldsymbol{x}, \boldsymbol{\epsilon})^{-1}\boldsymbol{y}$. We can then approximate equation (7) by replacing the integral by a maximum:

$$\int p_{\boldsymbol{E}}\big(\boldsymbol{\epsilon}(\boldsymbol{\alpha})\big) \frac{\mathcal{N}(\boldsymbol{\alpha} \,|\, \mathbf{0}, \mathbf{1})}{J(\boldsymbol{\alpha})} \, d\boldsymbol{\alpha} \approx \max_{\boldsymbol{\alpha}} p_{\boldsymbol{E}}\big(\boldsymbol{\epsilon}(\boldsymbol{\alpha})\big) \frac{\mathcal{N}(\boldsymbol{\alpha} \,|\, \mathbf{0}, \mathbf{1})}{J(\boldsymbol{\alpha})} = \max_{\boldsymbol{\epsilon}} p_{\boldsymbol{E}}(\boldsymbol{\epsilon}) \frac{\mathcal{N}(\boldsymbol{y} \,|\, \mathbf{0}, \boldsymbol{\Phi}\boldsymbol{\Phi}^T)}{J(\boldsymbol{\epsilon})}, \tag{8}$$

where in the last step we used the one-to-one correspondence between $\boldsymbol{\epsilon}$ and $\boldsymbol{\alpha}$.

After working out the details and taking the negative logarithm, the final optimization problem becomes:

$$-\log p(\boldsymbol{y}\,|\,\boldsymbol{x}) \approx \min_{\boldsymbol{\theta}_f,\boldsymbol{\epsilon}} \left( \underbrace{-\log p(\boldsymbol{\theta}_f)}_{\text{Hyperpriors}} \underbrace{-\log \mathcal{N}(\boldsymbol{\epsilon}\,|\,\boldsymbol{0},\boldsymbol{I})}_{\text{Noise prior}} \underbrace{-\log \mathcal{N}(\boldsymbol{y}\,|\,\boldsymbol{0},\boldsymbol{K})}_{\text{GP marginal}} + \underbrace{\sum_{i=1}^{N} \log \left| \boldsymbol{M}_{i\cdot}\boldsymbol{K}^{-1}\boldsymbol{y} \right|}_{\text{Information term}} \right).$$

$$(9)$$

Here, the kernel (Gram) matrix $\boldsymbol{K}$ is defined by $K_{ij} := k\big((x_i,\epsilon_i),(x_j,\epsilon_j)\big)$, where $k : \mathbb{R}^4 \to \mathbb{R}$ is the covariance function (5). It corresponds to $\boldsymbol{\Phi}\boldsymbol{\Phi}^T$ in our approximation. The matrix $\boldsymbol{M}$ contains the expected mean derivatives of the GP with respect to $e$ and is defined by $M_{ij} := \frac{\partial k}{\partial e}\big((x_i,\epsilon_i),(x_j,\epsilon_j)\big)$. Note that the matrices $\boldsymbol{K}$ and $\boldsymbol{M}$ both depend upon $\boldsymbol{\epsilon}$.

The Information term in the objective function (involving the partial derivatives $\frac{\partial k}{\partial e}$) may be surprising at first sight. It is necessary, however, to penalize dependences between $\boldsymbol{x}$ and $\boldsymbol{\epsilon}$: ignoring it would yield an optimal $\boldsymbol{\epsilon}$ that is heavily dependent on $\boldsymbol{x}$, violating assumption (B). Interestingly, this term is not present in the additive noise case that is usually considered, as the derivative of the causal mechanism with respect to the noise equals one, and its logarithm therefore vanishes. In the next subsection, we discuss some implementation issues that arise when one attempts to solve (6) and (9) in practice.

**Implementation issues**

First of all, we preprocess the observed data $\boldsymbol{x}$ and $\boldsymbol{y}$ by standardizing them to zero mean and unit variance for numerical reasons: if the length scales become too large, the kernel matrix $\boldsymbol{K}$ becomes difficult to handle numerically.

We solve the optimization problem (6) concerning the marginal distribution numerically by means of the algorithm written by Figueiredo and Jain [15]. We use a small but nonzero value ($10^{-4}$) of the regularization parameter.

The optimization problem (9) concerning the conditional distribution poses more serious practical problems. Basically, since we approximate a Bayesian integral by an optimization problem, the objective function (9) still needs to be regularized: if one of the partial derivatives $\frac{\partial f}{\partial e}$ becomes zero, the objective function diverges. In addition, the kernel matrix corresponding to (5) is extremely ill-posed. To deal with these matters, we propose the following *ad-hoc* solutions:

- We regularize the numerically ill-behaving logarithm in the last term in (9) by approximating it as $\log|x| \approx \log\sqrt{x^2 + \epsilon}$ with $\epsilon \ll 1$.
- We add a small amount of $\mathcal{N}(0,\sigma^2)$-uncertainty to each observed $y_i$-value, with $\sigma \ll 1$. This is equivalent to replacing $\boldsymbol{K}$ by $\boldsymbol{K} + \sigma^2\boldsymbol{I}$, which regularizes the ill-conditioned matrix $\boldsymbol{K}$. We used $\sigma = 10^{-5}$.

Further, note that in the final optimization problem (9), the unobserved noise values $\boldsymbol{\epsilon}$ can in fact also be regarded as additional hyperparameters, similar to the GPLVM model [16]. In our setting, this optimization is particularly challenging, as the number of parameters exceeds the number of observations. In particular, for small length scales $\lambda_X$ and $\lambda_E$ the objective function may exhibit a large number of local minima. In our implementation we applied the following measures to deal with this issue:

- We initialize $\boldsymbol{\epsilon}$ with an additive noise model, by taking the residuals from a standard GP regression as initial values for $\boldsymbol{\epsilon}$. The reason for doing this is that in an additive noise model, all partial derivatives $\frac{\partial f}{\partial e}$ are positive and constant. This initialization effectively leads to a solution that satisfies the invertability assumption that we made in approximating the evidence.[5]
- We implemented a log barrier that heavily penalized negative values of $\frac{\partial f}{\partial e}$. This was done to avoid sign flips of these terms that would violate the invertability assumption. Basically, together with our earlier regularization of the logarithm, we replaced the logarithms $\log|x|$ in

the last term in (9) by:

$$\log \sqrt{(x-\epsilon)^2 + \epsilon} + A\big(\log\sqrt{(x-\epsilon)^2+\epsilon} - \log\sqrt{\epsilon}\big)\mathbb{1}_{x\leq\epsilon}$$

with $\epsilon \ll 1$. We used $\epsilon = 10^{-3}$ and $A = 10^2$.

The resulting optimization problem can be solved using standard numerical optimization methods (we used LBFGS). The source code of our implementation is available as supplementary material and can also be downloaded from `http://webdav.tuebingen.mpg.de/causality/`.

## 3   Experiments

To evaluate the ability of our method to identify causal directions, we have tested our approach on simulated and real-world data. To identify the most probable causal direction, we evaluate the marginal likelihoods corresponding to both possible causal directions (which are given by combining the results of equations (6) and (9)), choosing the model that assigns higher probability to the observed data. We henceforth refer to this approach as *GPI-MML*. For comparison, we also considered the marginal likelihood using a GP covariance function that is constant with respect to $e$, i.e., assuming additive noise. For this special case, the noise values $e$ can be integrated out analytically, resulting in standard GP regression. We call this approach *AN-MML*. We also compare with the method proposed in [1], which also uses an additive noise GP regression for the conditional model, but uses a simple Gaussian model for the input distribution $p(x)$. We refer to this approach as *AN-GAUSS*.

We complemented the marginal likelihood as selection criterion with another possible criterion for causal model selection: the independence of the cause and the estimated noise [5]. Using HSIC [17] as test criterion for independence, this approach can be applied to both the additive noise GP and the more general latent variable approach. As the marginal likelihood does not provide a significance level for the inferred causal direction, we used the ratio of the $p$-values of HSIC for both causal directions as prediction criterion, preferring the direction with a higher $p$-value (i.e., with less dependence between the estimated noise and the cause). HSIC as selection criterion applied to the additive or general Gaussian process model will be referred to as *AN-HSIC* and *GPI-HSIC* respectively.

We compared these methods with other related methods: *IGCI* [13], a method that is also based on assumption (C), although designed for the noise-free case; *LINGAM* [2], which assumes a linear causal mechanism; and *PNL*, the Post-NonLinear model [6]. We evaluated all methods in the "forced decision" scenario, i.e., the only two possible decisions that a method could take were $X \to Y$ and $Y \to X$ (so decisions like "both models fit the data" or "neither model fits the data" were not possible).

**Simulated data**   Inspired by the experimental setup in [5], we generated simulated datasets from the model $Y = (X + bX^3)e^{\alpha E} + (1-\alpha)E$. Here, the random variables $X$ and $E$ where sampled from a Gaussian distribution with their absolute values raised to the power $q$, while keeping the original sign. The parameter $\alpha$ controls the type of the observation noise, interpolating between purely additive noise ($\alpha = 0$) and purely multiplicative noise ($\alpha = 1$). The coefficient $b$ determines the non-linearity of the true causal model, with $b = 0$ corresponding to the linear case. Finally, the parameter $q$ controls the non-Gaussianity of the input and noise distributions: $q = 1$ gives a Gaussian, while $q > 1$ and $q < 1$ produces super-Gaussian and sub-Gaussian distributions respectively.

For alternative parameter settings $\alpha, b$ and $q$, we generated $D = 40$ independent datasets. Each dataset consisted of $N = 500$ samples from the corresponding generative model. Figure 2 shows the accuracy of the considered methods evaluated on these simulated datasets. Encouragingly, GPI appears to be robust with respect to the type of noise, outperforming additive noise models in the full range between additive and multiplicative noise (Figure 2a). Note that the additive noise models actually yield the *wrong* decision for high values of $\alpha$, whereas the GPI methods stay well above chance level. Figure 2b shows accuracies for a linear model and a non-Gaussian noise and input distribution. Figure 2c shows accuracies for a non-linear model with Gaussian additive noise. We observe that GPI-MML performs well in each scenario. Further, we observe that AN-GAUSS, the method proposed in [1], only performs well for Gaussian input distributions and additive noise.

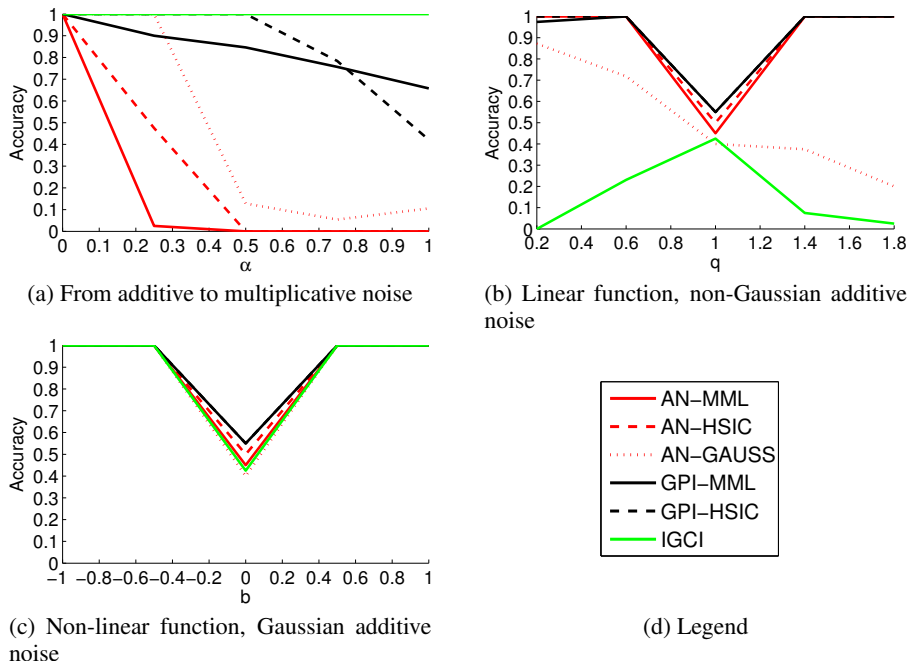

(a) From additive to multiplicative noise

(b) Linear function, non-Gaussian additive noise

(c) Non-linear function, Gaussian additive noise

(d) Legend

Figure 2: Accuracy of recovering the true causal direction in simulated datasets. **(a)** From additive ($\alpha = 0$) to multiplicative noise ($\alpha = 1$), for $q = 1$ and $b = 1$; **(b)** from sub-Gaussian noise ($q < 1$), Gaussian noise ($q = 1$) to super-Gaussian noise ($q > 1$), for a linear function ($b = 0$) with additive noise ($\alpha = 0$); **(c)** from non-linear ($b < 0$) to linear ($b = 0$) to non-linear ($b > 1$), with additive Gaussian noise ($q = 1, \alpha = 0$).

Table 1: Accuracy (in percent) of recovering the true causal direction in 68 real world datasets.

| AN-MML | AN-HSIC | AN-GAUSS | GPI-MML | GPI-HSIC | IGCI | LINGAM | PNL |
|--------|---------|----------|---------|----------|------|--------|-----|
| $68 \pm 1$ | $68 \pm 3$ | $45 \pm 3$ | $72 \pm 2$ | $62 \pm 4$ | $76 \pm 1$ | $62 \pm 3$ | $67 \pm 4$ |

**Results on cause-effect pairs** Next, we applied the same methods and selection criteria to real-world cause-effect pairs where the true causal direction is known. The data was obtained from `http://webdav.tuebingen.mpg.de/cause-effect/`. We considered a total of 68 pairs in this dataset collected from a variety of domains. To reduce computation time, we subsampled the data, using a total of at most $N = 500$ samples for each cause-effect pair. Table 1 shows the prediction accuracy for the same approaches as in the simulation study, reporting averages and standard deviations estimated from 3 repetitions of the experiments with different subsamples.

## 4 Conclusions and discussion

We proposed the first method (to the best of our knowledge) for addressing the challenging task of distinguishing between cause and effect *without* an *a priori* restriction to a certain class of models. The method compares marginal likelihoods that penalize complex input distributions and causal mechanisms. Moreover, our framework generalizes a number of existing approaches that assume a limited class of possible causal mechanisms functions. A more extensive evaluation of the performance of our method has to be performed in future. Nevertheless, the encouraging results that we have obtained thus far confirm the hypothesis that asymmetries of the joint distribution of cause and effect provide useful hints on the causal direction.

**Acknowledgments**

We thank Stefan Harmeling and Hannes Nickisch for fruitful discussions. We also like to thank the authors of the GPML toolbox [18], which was very useful during the development of our software. OS was supported by a fellowship from the Volkswagen Foundation.

## Footnotes

[1]For the special case of additive Gaussian noise, the method proposed in [1] would also seem to be a valid Bayesian approach to causal discovery with continuous variables. However, that approach is flawed, as it either completely ignores the distribution for the cause, or uses a simple Gaussian marginal distribution for the cause, which may not be realistic (from the paper it is not clear exactly what is proposed). But, as suggested by Theorem 1, and as illustrated by our empirical results, the complexity of the input distribution plays an important role here that cannot be neglected, especially in the two-variable case.

[2]This assumption may be violated in biological systems, for example, where the causal mechanisms may have been tuned to their input distributions through evolution.

[3]This is not a restriction of the model class, since in general we can write $E = g(\bar{E})$ for some function $g$, with $\bar{E} \sim \mathcal{N}(0, 1)$ and $\bar{f} = f(\cdot, g(\cdot))$.

[4]Alternatively, one could first integrate over the causal mechanisms $f$, and then optimize over the noise values $\boldsymbol{e}$, similar to what is usually done in GPLVMs [16]. However, we believe that for the purpose of causal discovery, that approach does not work well. The reason is that when optimizing over $\boldsymbol{e}$, the result is often quite dependent on $\boldsymbol{x}$, which violates our basic assumption that $X \perp\!\!\!\perp E$. The approach we follow here is more related to nonlinear ICA, whereas GPLVMs are related to nonlinear PCA.

[5]This is related in spirit to the standard initialization of GPLVM models by PCA.

# References

[1] N. Friedman and I. Nachman. Gaussian process networks. In *Proc. of the 16th Annual Conference on Uncertainty in Artificial Intelligence*, pages 211–219, 2000.

[2] S. Shimizu, P. O. Hoyer, A. Hyvärinen, and A. J. Kerminen. A linear non-Gaussian acyclic model for causal discovery. *Journal of Machine Learning Research*, 7:2003–2030, 2006.

[3] X. Sun, D. Janzing, and B. Schölkopf. Causal inference by choosing graphs with most plausible Markov kernels. In *Proceeding of the 9th Int. Symp. Art. Int. and Math.*, Fort Lauderdale, Florida, 2006.

[4] X. Sun, D. Janzing, and B. Schölkopf. Distinguishing between cause and effect via kernel-based complexity measures for conditional probability densities. *Neurocomputing*, pages 1248–1256, 2008.

[5] P. O. Hoyer, D. Janzing, J. M. Mooij, J. Peters, and B. Schölkopf. Nonlinear causal discovery with additive noise models. In D. Koller, D. Schuurmans, Y. Bengio, and L. Bottou, editors, *Advances in Neural Information Processing Systems 21 (NIPS*2008)*, pages 689–696, 2009.

[6] K. Zhang and A. Hyvärinen. On the identifiability of the post-nonlinear causal model. In *Proceedings of the 25th Conference on Uncertainty in Artificial Intelligence*, Montreal, Canada, 2009.

[7] D. Janzing, P. Hoyer, and B. Schölkopf. Telling cause from effect based on high-dimensional observations. In *Proceedings of the International Conference on Machine Learning (ICML 2010)*, pages 479–486, 2010.

[8] J. M. Mooij and D. Janzing. Distinguishing between cause and effect. In *Journal of Machine Learning Research Workshop and Conference Proceedings*, volume 6, pages 147–156, 2010.

[9] P. Spirtes, C. Glymour, and R. Scheines. *Causation, Prediction, and Search*. Springer-Verlag, 1993. (2nd ed. MIT Press 2000).

[10] J. Pearl. *Causality: Models, Reasoning, and Inference*. Cambridge University Press, 2000.

[11] J. Lemeire and E. Dirkx. Causal models as minimal descriptions of multivariate systems. http://parallel.vub.ac.be/~jan/, 2006.

[12] D. Janzing and B. Schölkopf. Causal inference using the algorithmic Markov condition. *IEEE Transactions on Information Theory*, 56(10):5168–5194, 2010.

[13] P. Daniušis, D. Janzing, J. M. Mooij, J. Zscheischler, B. Steudel, K. Zhang, and B. Schölkopf. Inferring deterministic causal relations. In *Proceedings of the 26th Annual Conference on Uncertainty in Artificial Intelligence (UAI-10)*, 2010.

[14] A. Hyvärinen and P. Pajunen. Nonlinear independent component analysis: Existence and uniqueness results. *Neural Networks*, 12(3):429–439, 1999.

[15] M. A. T. Figueiredo and A. K. Jain. Unsupervised learning of finite mixture models. *IEEE Transactions on Pattern Analysis and Machine Intelligence*, 24(3):381–396, March 2002.

[16] N. D. Lawrence. Gaussian process latent variable models for visualisation of high dimensional data. In *Advances in Neural Information Processing Systems 16: Proceedings of the 2003 Conference*, page 329. The MIT Press, 2004.

[17] A. Gretton, R. Herbrich, A. Smola, O. Bousquet, and B. Schölkopf. Kernel methods for measuring independence. *Journal of Machine Learning Research*, 6:2075–2129, 2005.

[18] C. E. Rasmussen and H. Nickisch. Gaussian Processes for Machine Learning (GPML) Toolbox. *Journal of Machine Learning Research*, accepted, 2010.

